# Inverse M-Kernels for Linear Universal Approximators of Non-Negative Functions

**Hideaki Kim**
NTT Corporation
`hideaki.kin@ntt.com`

## Abstract

Kernel methods are widely utilized in machine learning field to learn, from training data, a latent function in a reproducing kernel Hilbert space. It is well known that the approximator thus obtained usually achieves a linear representation, which brings various computational benefits, while maintaining great representation power (i.e., universal approximation). However, when non-negativity constraints are imposed on the function's outputs, the literature usually takes the kernel method-based approximators as offering linear representations at the expense of limited model flexibility or good representation power by allowing for their nonlinear forms. The main contribution of this paper is to derive a sufficient condition for a positive definite kernel so that it may construct flexible and linear approximators of non-negative functions. We call a kernel function that offers these attributes an *inverse M-kernel*; it is a generalization of the inverse M-matrix. Furthermore, we show that for a one-dimensional input space, universal exponential/Abel kernels are inverse M-kernels and construct linear universal approximators of non-negative functions. To the best of our knowledge, it is the first time that the existence of linear universal approximators of non-negative functions has been elucidated. We confirm the effectiveness of our results by experiments on the problems of non-negativity-constrained regression, density estimation, and intensity estimation. Finally, we discuss issues and perspectives on multi-dimensional input settings.

## 1 Introduction

Non-parametric estimation of latent functions continues to be of theoretical and practical importance in a wide spectrum of disciplines such as signal/image processing [13, 32], system control [12], geostatistics [3], bioinformatics [28], and clinical research [4]. Kernel method, one of the most established techniques, learns flexible function approximators by embedding data points into higher dimensional reproducing kernel Hilbert spaces (RKHSs) [26, 29]. For a broad class of learning problems, kernel methods invoke the representer theorem [27, 35] and recast the infinite-dimensional functional problems as their finite-dimensional counterparts, where the obtained approximators have linear representation, i.e., finite linear combinations of kernel functions evaluated on data points. Significant computational benefits are attained by the linear representation such as convex optimization and cheap evaluation/integration of approximators.

In recent years, great attention has been paid to kernel methods with non-negativity constraints on function outputs [6, 17, 17, 31]; crucial applications include non-negativity-constrained regression, density estimation, and intensity estimation. Compared to unconstrained alternatives, non-negativity-constrained kernel methods developed to date are faced with a problematic trade-off between linearity and flexibility: the obtained approximators either can have linear representations at the expense of degraded representation power [17], or achieve good representation power (i.e., uni-

versal approximation) by accepting nonlinear forms [17, 21], which incurs substantial computation costs. To the best of our knowledge, no non-negativity-constrained kernel method has been proposed that combines linear representation with good representation power.

In this paper, we propose the first linear universal approximator of non-negative functions for one-dimensional input spaces. First, we derive a sufficient condition so that the kernel can construct a linear approximator of a non-negative function. We call a kernel that satisfies this novel condition an *inverse M-kernel*; it is a generalization of inverse M-matrix [8]. Next, we show that exponential/Abel kernels, which have the universal approximating property [19, 30], are inverse M-kernel functions and can construct linear universal approximators of non-negative functions for one-dimensional input spaces. It is worth noting that the most popular Gaussian kernels do not satisfy the condition demanded by the inverse M-kernel. Our results shed light on exponential kernels, which have received less attention in the literature, as universal kernels for non-negativity-constrained approximators on one-dimensional input spaces.

In Section 2, we outline related works and introduce some known results on M-matrix theory used throughout the paper. In Section 3, we introduce the inverse M-kernel and construct linear universal approximators of non-negative functions. In Section 4, we show some important applications of our results, which include non-negativity-constrained regression, density estimation, and intensity estimation, and evaluate the effectiveness of our proposal on synthetic data[1]. Finally, Section 5 states our conclusions and discuss future works on multi-dimensional input settings.

## 2 Background

### 2.1 Kernel Method-Based Linear Approximator

Let $\mathcal{X}$ be a prescribed input space and $k : \mathcal{X} \times \mathcal{X} \to \mathbb{R}$ be a positive semi-definite kernel. Then there exists a unique reproducing kernel Hilbert space (RKHS) $\mathcal{H}_k$ [26, 29] associated with kernel $k(\cdot, \cdot)$. Given a set of $N$ points $\{x_n \in \mathcal{X}\}_{n=1}^N$ and a regularized learning problem:

$$\min_{f \in \mathcal{H}_k} L(f(x_1), \ldots, f(x_N)) + \Omega(||f||^2_{\mathcal{H}_k}),\tag{1}$$

where $L : \mathbb{R}^N \to \mathbb{R}$ is a loss function, and $\Omega$ is a non-decreasing function of the squared RKHS norm of $f$. It is well known that the solution of (1) invokes the *representer theorem* [27, 35] and has the representation,

$$f^*(\cdot) = \sum_{n=1}^N \alpha_n k(x_n, \cdot).\tag{2}$$

For simplicity, the linear regularizer, $\Omega(||f||^2_{\mathcal{H}_k}) = r||f||^2_{\mathcal{H}_k}$ for $r \geq 0$, is assumed in this paper. The infinite-dimensional optimization problem (1) can be reduced to a finite-dimensional one in terms of coefficients $\alpha := (\alpha_1, \ldots, \alpha_N)^\top \in \mathbb{R}^N$ as follows:

$$\min_{\alpha \in \mathbb{R}^N} L(\boldsymbol{K}\alpha) + r\, \alpha^\top \boldsymbol{K}\alpha, \qquad \boldsymbol{K} := [k(x_n, x_{n'})]_{nn'} \in \mathbb{R}^{N \times N}.\tag{3}$$

Under universal kernels [19, 30], where RKHSs are dense in the space of all continuous functions of $\mathcal{X}$, the linear approximator (2) can approximate any continuous function on $\mathcal{X}$, which is a property known as the *universal approximation*. Each evaluation of the objective function (3) and the linear approximator (2) needs the computation of $\mathcal{O}(N^2)$ and $\mathcal{O}(N)$, respectively.

When non-negativity constraints are imposed on target function $f \geq 0$, the literature considers that the linear approximator (2) cannot be applied directly because it generally has negative values, even under non-negative kernel $k(\cdot, \cdot) \geq 0$. Conventional approaches to address the problem are listed below (Section 2.2-2.3).

## 2.2 Linear Approximators with Non-Negativity Constraints

Non-negative coefficients Model (NCM) [17] enforces the linear approximator (2) to be non-negative by using non-negative coefficients and kernel functions,

$$f_{\text{NCM}}(\cdot) = \sum_{n=1}^{N} \alpha_n k(x_n, \cdot), \qquad \alpha_1, \dots, \alpha_N \geq 0, \quad k(\cdot, \cdot) \geq 0, \tag{4}$$

where coefficients $\alpha \in \mathbb{R}^N$ are obtained by solving the optimization problem (3) with constraint $\alpha_n \geq 0$. Here non-negative kernels include popular kernels such as Gaussian kernels $e^{-||x-x'||^2}$, exponential kernels $e^{-||x-x'||}$, and Cauchy kernels $(1 + ||x - x'||^2)^{-1}$. Although NCM enjoys great computational benefits from its linear representation such as the preservation of loss functional's convexity and cheap evaluations/integrations, it suffers from low representation power due to the strong non-negativity constraint on coefficients, $\alpha_n \geq 0$ (for details, see Appendix B).

Note that linear approaches with partial non-negativity constraints [5, 17, 22], which require non-negativity only on a finite number of points (the points are not necessarily data points $\{x_n\}_{n=1}^N$), do not guarantee the non-negativity at locations other than the points; these approaches are out of scope of this paper.

## 2.3 Nonlinear Approximators with Non-Negativity Constraints

An elegant quadratic form of the non-negative model (QNM) [17] has recently been proposed that exploits the non-negativity inherent in positive semi-definite operators:

$$f_{\text{QNM}}(\cdot) = \sum_{n,n'=1}^{N} B_{nn'} k(x_n, \cdot) k(x_{n'}, \cdot), \qquad B \in \mathbb{R}^{N \times N}, \quad B \succeq 0, \tag{5}$$

where $\succeq 0$ represents the positive semi-definite constraint. QNM has the following beneficial properties: it preserves the convexity of the loss functionals; it can be integrated in a closed form if we know how to integrate kernel functions; under mild conditions on kernels, it is a universal approximator for non-negative functions. The coefficient matrix $B$ is obtained efficiently by solving an $N$-dimensional optimization problem, which naively costs $\mathcal{O}(N^3)$ for each evaluation of the objective function (for details, see Appendix A). An evaluation of the approximator (5) needs the computation of $\mathcal{O}(N^2)$.

Generalized linear models (GLMs), another nonlinear approach to constructing a non-negative function, use nonlinear transformation of a linear model [21]. GLMs are so flexible that they can represent a wide class of non-negative functions, but they generally do not preserve the convexity of loss functionals where they are used and cannot be integrated in closed form. However, most recently, a promising model called squared neural family (SNF) [34], which was specifically designed for density estimation, has been proposed that uses a quadratic transformation,

$$f_{\text{SNF}}(\cdot) dx = \frac{d\mu(\cdot)}{z(\Theta)} ||g(t(\cdot); \Theta)||^2, \quad g(t; \Theta) = V\sigma(Wt + b), \quad \Theta = (V, W, b), \tag{6}$$

where $d\mu(\cdot)$ is a non-negative measure, $g(\cdot|\Theta)$ is a neural network of one hidden layer with activation function $\sigma(\cdot)$ and parameter $\Theta$, $t(\cdot)$ is a sufficient statistic, and $z(\Theta)$ is the normalizing constant, $z(\Theta) = \int_{\mathcal{X}} ||g(t(x); \Theta)||^2 d\mu(x)$. Thanks to the integrations of SNF (i.e., $z(\Theta)$) can be executed in a closed form under various $d\mu(\cdot), t(\cdot)$, and $\sigma(\cdot)$. But SNF has some possible drawbacks: it cannot preserve the convexity of the loss functionals, and so can yield many local optima; it has many hyper-parameters to be determined such as $\sigma(\cdot)$ and the size of parameter $\Theta$, on which model performance largely depends. In this paper, we adopted the same size of $\Theta$ as in [34]: $V \in \mathbb{R}^{1 \times 30}$, $W \in \mathbb{R}^{30 \times dim(\mathcal{X})}, b \in \mathcal{R}^{30}$, and $t(x) = x$.

## 2.4 M-Matrix Theory

Here we introduce some existing results on M-matrix theory, which are then used to clarify a sufficient kernel condition so that it may construct a linear approximator of non-negative functions.

The $n$-by-$n$ real matrix $\boldsymbol{A} \in \mathbb{R}^{n \times n}$ is called an *M-matrix* [23] if it has the form $\gamma \boldsymbol{I}_n - \boldsymbol{C}$, in which $\boldsymbol{I}_n \in \mathbb{R}^{n \times n}$ is the identity matrix of size $n$, $\boldsymbol{C} \in \mathbb{R}^{n \times n}$ is an entry-wise non-negative matrix, and $\gamma > \rho(\boldsymbol{C})$, the spectral radius of $\boldsymbol{C}$; this is equivalent to $\boldsymbol{A}$ with non-positive off-diagonal entries that is invertible and having an entry-wise non-negative inverse. An entry-wise non-negative matrix that occurs as the inverse of an M-matrix is called an *inverse M-matrix* [8]. Note that the inverse of an M-matrix is always entry-wise non-negative, while the reverse is not always true. We denote the classes of M-matrices and inverse M-matrices by $\mathcal{M}$ and $\mathcal{M}^{-1}$, respectively. Lemma 1 below shows useful properties on the entry-wise sign patterns of partitioned inverse M-matrices (for the proof, see Theorem 8 in [8]).

**Lemma 1.** *Suppose that an $n$-by-$n$ symmetric inverse M-matrix $\boldsymbol{Q} \in \mathcal{M}^{-1}$ is partitioned as*

$$\boldsymbol{Q} = \begin{pmatrix} \boldsymbol{Q}_1 & \boldsymbol{Q}_3^\top \\ \boldsymbol{Q}_3 & \boldsymbol{Q}_2 \end{pmatrix}, \qquad \boldsymbol{Q}_1 \in \mathbb{R}^{n_1 \times n_1},\ \boldsymbol{Q}_2 \in \mathbb{R}^{n_2 \times n_2},\ \boldsymbol{Q}_3 \in \mathbb{R}^{n_2 \times n_1},$$

*where $n_1$ and $n_2$ are positive integers satisfying $n = n_1 + n_2$. Then the following inequalities hold:*

$$0 \leqslant \boldsymbol{Q}_3^\top \boldsymbol{Q}_2^{-1}, \qquad 0 \leqslant \boldsymbol{Q}_1 - \boldsymbol{Q}_3^\top \boldsymbol{Q}_2^{-1} \boldsymbol{Q}_3 \in \mathcal{M}^{-1},$$

*where $\leqslant (\geqslant)$ represents the entry-wise inequality of matrices/vectors.*

*Proof.* Suppose that the inverse of $\boldsymbol{Q}$, denoted by $\bar{\boldsymbol{Q}} \in \mathcal{M}$, is partitioned as

$$\bar{\boldsymbol{Q}} = \begin{pmatrix} \bar{\boldsymbol{Q}}_1 & \bar{\boldsymbol{Q}}_3^\top \\ \bar{\boldsymbol{Q}}_3 & \bar{\boldsymbol{Q}}_2 \end{pmatrix}, \qquad \bar{\boldsymbol{Q}}_1 \in \mathbb{R}^{n_1 \times n_1},\ \bar{\boldsymbol{Q}}_2 \in \mathbb{R}^{n_2 \times n_2},\ \bar{\boldsymbol{Q}}_3 \in \mathbb{R}^{n_2 \times n_1}.$$

Then $\bar{\boldsymbol{Q}}_1$ and $\bar{\boldsymbol{Q}}_3^\top$ may be expressed by using Schur's complements as

$$\bar{\boldsymbol{Q}}_1 = (\boldsymbol{Q}_1 - \boldsymbol{Q}_3^\top \boldsymbol{Q}_2^{-1} \boldsymbol{Q}_3)^{-1}, \quad \bar{\boldsymbol{Q}}_3^\top = -(\boldsymbol{Q}_1 - \boldsymbol{Q}_3^\top \boldsymbol{Q}_2^{-1} \boldsymbol{Q}_3)^{-1} \boldsymbol{Q}_3^\top \boldsymbol{Q}_2^{-1}.$$

Because $\bar{\boldsymbol{Q}}_1$, a principal submatrix of M-matrix $\bar{\boldsymbol{Q}}$, is an M-matrix (Corollary 3 in [8]), $\boldsymbol{Q}_1 - \boldsymbol{Q}_3^\top \boldsymbol{Q}_2^{-1} \boldsymbol{Q}_3 = \bar{\boldsymbol{Q}}_1^{-1}$ is an inverse M-matrix and entry-wise positive. Furthermore, $\boldsymbol{Q}_3^\top \boldsymbol{Q}_2^{-1} = -(\boldsymbol{Q}_1 - \boldsymbol{Q}_3^\top \boldsymbol{Q}_2^{-1} \boldsymbol{Q}_3) \bar{\boldsymbol{Q}}_3^\top$ is an entry-wise positive matrix because $(\boldsymbol{Q}_1 - \boldsymbol{Q}_3^\top \boldsymbol{Q}_2^{-1} \boldsymbol{Q}_3) \in \mathcal{M}^{-1}$ and $-\bar{\boldsymbol{Q}}_3^\top$ are both entry-wise positive matrices. This completes the proof. ∎

## 3 Inverse M-Kernels

In this section, we define a new class of kernels and show that it plays an essential role in constructing a linear and flexible approximator of non-negative functions.

**Definition 1** (Inverse M-kernels). *Let $k : \mathcal{X} \times \mathcal{X} \to \mathbb{R}_+$ be a positive semi-definite kernel that outputs non-negative values, $\boldsymbol{K} := [k(x_n, x_{n'})]_{nn'}$ be a gram matrix constructed for any set of $N$ points $(x_1, x_2, \ldots, x_N)$, and $s : \mathbb{N} \to \mathbb{R}_+$ be a non-negative function of data size $N$. We call $k(\cdot, \cdot)$ an inverse M-kernel if $\boldsymbol{K} + s(N)\boldsymbol{I}_N \in \mathcal{M}^{-1}$. We denote the class of inverse M-kernels by $\mathcal{F}_{\mathcal{M}^{-1}}^{s(N)}$. Also, if $s(N) = 0$ or $\boldsymbol{K} \in \mathcal{M}^{-1}$, we call the kernel a strict inverse M-kernel, which we denote by $k(\cdot, \cdot) \in \mathcal{F}_{\mathcal{M}^{-1}}$.*

The non-negative function $s(N)$ may generally exhibit various scalings with respect to $N$, but as will be discussed later, smaller scalings offer greater advantages. In this paper, we will focus solely on examples with scalings of $\mathcal{O}(1)$ and $\mathcal{O}(N)$.

### 3.1 Inverse M-Kernel Models

We now consider the following linear approximator with an inverse M-kernel:

$$f_{\text{IMK}}(\cdot) = \sum_{n=1}^{N} \alpha_n k(x_n, \cdot) = \boldsymbol{k}(\cdot)^\top \alpha, \quad (\boldsymbol{K} + s(N+1)\boldsymbol{I}_N)\alpha \geqslant 0,\ k(\cdot, \cdot) \in \mathcal{F}_{\mathcal{M}^{-1}}^{s(N)}, \quad (7)$$

where $\{x_n\}_{n=1}^{N}$ is the $N$ data points, $\alpha := (\alpha_1, \ldots, \alpha_N)^\top$, and $\boldsymbol{k}(x) := (k(x_1, x), \ldots, k(x_N, x))^\top$. Coefficients $\alpha$ are obtained by solving the optimization problem (3) with constraint $(\boldsymbol{K} + s(N + 1)\boldsymbol{I}_N)\alpha \geqslant 0$. We call this approximator an *inverse M-kernel model* (IMK). Then Theorem 1 below guarantees the non-negativity of the approximator (7).

**Theorem 1** (Non-negativity of inverse M-kernel models). *The inverse M-kernel models $f_{IMK}(x)$ defined by (7) are non-negative for any input point $x \in \mathcal{X}$.*

*Proof.* Consider the $(N+1)$-by-$(N+1)$ gram matrix for the data points $\{x_n\}_{n=1}^N$ and any point $x \in \mathcal{X}$ such that

$$\boldsymbol{Q} = \begin{pmatrix} k(x,x) & \boldsymbol{k}(x)^\top \\ \boldsymbol{k}(x) & \boldsymbol{K} \end{pmatrix} + s(N+1)\boldsymbol{I}_{N+1}.$$

Gram matrix $Q$ is an inverse M-matrix because $k(\cdot, \cdot) \in \mathcal{F}_{\mathcal{M}^{-1}}^{s(N)}$, and according to the first inequality in Lemma 1, the following relation holds: $\boldsymbol{k}(x)^\top (\boldsymbol{K} + s(N+1)\boldsymbol{I}_N)^{-1} \geqslant 0$. Let $\beta \in \mathbb{R}_+^N$ be a non-negative vector, $\beta \geqslant 0$, then the following inner product of non-negative vectors completes the proof: $\boldsymbol{k}(x)^\top (\boldsymbol{K} + s(N+1)\boldsymbol{I}_N)^{-1}\beta = \boldsymbol{k}(x)^\top \alpha \geq 0$, for $\beta = (\boldsymbol{K} + s(N+1)\boldsymbol{I}_N)\alpha \geqslant 0$. ∎

It should be emphasized here that the comparison between the two linear models, $f_{IMK}$ in (7) and $f_{NCM}$ in (4), suggests that our proposed $f_{IMK}$ should have substantially greater representation power than $f_{NCM}$: $f_{IMK}$'s constraint on coefficient, $(\boldsymbol{K} + s(N+1)\boldsymbol{I}_N)\alpha \geqslant 0$, is much weaker than $f_{NCM}$'s, $\alpha \geqslant 0$. Here, the discrepancy between the two models is controlled by $s(N+1)$, and $f_{IMK}$ reduces to $f_{NCM}$ for $s(N+1) \to \infty$: $(\boldsymbol{K}+s(N+1)\boldsymbol{I}_N)\alpha \geqslant 0 \Leftrightarrow (s(N+1)^{-1}\boldsymbol{K}+\boldsymbol{I}_N)\alpha \geqslant 0 \overset{s \to \infty}{=} \alpha \geqslant 0$. Clearly, $f_{IMK}$ has greatest representation power when $s(N+1)$ is equal to zero, and we can derive a sufficient condition on $f_{IMK}$ so that it may be a universal approximator of non-negative function.

**Theorem 2** (Condition for linear universal approximation). *The inverse M-kernel model $f_{IMK}$ defined by (7) is a universal approximator of non-negative functions if the kernel is universal and a strict inverse M-kernel.*

*Proof.* Let $k : \mathcal{X} \times \mathcal{X} \to \mathbb{R}$ be a universal kernel, and let $\mathcal{Z}$ be a compact subset of $\mathcal{X}$. Then the corresponding RKHS is equal to the space of all continuous functions from $\mathcal{Z}$, denoted by $\mathcal{C}(\mathcal{Z})$, which is equipped with maximum norm $||\cdot||_{\mathcal{C}(\mathcal{Z})}$. Suppose that we have a set of data points, $\{(x_n, g(x_n))\}_{n=1}^N$, for non-negative target function $g : \mathcal{Z} \to \mathbb{R}_+$ in $\mathcal{C}(\mathcal{Z})$, and $k(\cdot, \cdot)$ is a strict inverse M-kernel $k(\cdot, \cdot) \in \mathcal{F}_{\mathcal{M}^{-1}}$. Then we can rewrite the inverse M-kernel model (7) in the form of noise-free kernel ridge regression (KRR) as:

$$f_{IMK}(x) = \boldsymbol{k}(x)^\top \boldsymbol{K}^{-1}\tilde{g}, \qquad \tilde{g} := (g(x_1), \ldots, g(x_N))^\top \geqslant 0.$$

Because constraint $\tilde{g} \geqslant 0$ is satisfied for any $\{x_n\}_{n=1}^N$ due to the non-negativity of $g(\cdot)$, we can apply the generalization error bound of a normal KRR (Proposition 1 in [15]) to it:

$$||f_{IMK}(\cdot) - g(\cdot)||_{\mathcal{C}(\mathcal{Z})} < \sup_{x \in \mathcal{Z}} \sqrt{k(x,x) - \boldsymbol{k}(x)^\top \boldsymbol{K}^{-1}\boldsymbol{k}(x)} \cdot \Gamma,$$

where $\Gamma$ is a constant. The upper bound goes to zero if $N \to \infty$ and $\{x_n\}_{n=1}^N$ is aligned appropriately, indicating that given $\epsilon > 0$, there exists $\{x_n\}_{n=1}^N$ such that $||f_{IMK}(\cdot) - g(\cdot)||_{\mathcal{C}(\mathcal{Z})} \leq \epsilon$, which completes the proof. ∎

### 3.2 Equivalent Inverse M-Kernels for Permanental Processes

As a by-product of inverse M-kernels, we can address a well-known *nodal line problem* [7] on Poisson intensity estimation with reproducing kernels [6]: Given a set of $N$ points $\{x_n\}_{n=1}^N$ observed for compact domain $\mathcal{T}$, intensity function $\lambda(x)$, an instantaneous probability of events occurring at each point on $\mathcal{T}$, is estimated by a linear model with the equivalent kernel function $h(\cdot, \cdot)$ so that

$$\lambda(x) = f^2(x), \quad f(x) = \sum_{n=1}^N h(x, x_n)v_n^*, \quad v^* = \arg\min_{v \in \mathbb{R}^N} -\sum_{n=1}^N \log f(x_n) + r\, v^\top \boldsymbol{H}v, \quad (8)$$

where $\boldsymbol{H} := [h(x_n, x_{n'})]_{nn'}$, and $h(\cdot, \cdot)$ solves an integral equation constructed by kernel function $k(\cdot, \cdot)$ as $h(x, x') + 2/r \int_{\mathcal{T}} k(x,s)h(s,x')ds = k(x,x')$; $f(\cdot)$ generally may have negative values, which causes many local modes since $\pm f(\cdot)$ can lead to similar intensity $\lambda(\cdot) = f^2(\cdot)$, resulting in artificial zero crossings of $f(\cdot)$, especially on locations where the intensity is low. If $h(\cdot, \cdot)$ is a strict inverse M-kernel, then the linear model of $f(\cdot)$ can constitute an inverse M-kernel model (7), which is non-negative at any $x \in \mathcal{T}$ under a weak constraint of coefficient, $\boldsymbol{H}v \geqslant 0$.

We now consider solving the integral equation for $h(\cdot, \cdot)$ with the naive approach [10], which approximates the integral operator by $J$-point numerical integration, resulting in

$$h(x, x') = k(x, x') - \boldsymbol{k}_J(x)^\top \big(w\boldsymbol{I}_J + \boldsymbol{K}_J\big)^{-1}\boldsymbol{k}_J(x'), \quad w = rJ/(2|\mathcal{T}|), \qquad (9)$$

where $\boldsymbol{k}_J(x) := (k(x, q_1), \dots, k(x, q_J))^\top$, $\boldsymbol{K}_J := [k(q_j, q_{j'})]_{jj'}$, $|\mathcal{T}| := \int_\mathcal{T} dx$, and $(q_1, \dots, q_J)$ is the regularly aligned evaluation points. Theorem 3 below shows a sufficient condition on the equivalent kernel such that it may be a strict inverse M-kernel.

**Theorem 3** (Equivalent inverse M-kernels). *The equivalent kernel $h(\cdot, \cdot)$ defined by (9) is a strict inverse M-kernel if the corresponding kernel $k(\cdot, \cdot)$ is a strict inverse M-kernel.*

*Proof.* Consider the $(J+2)$-by-$(J+2)$ gram matrix for the points $\{q_j\}_{j=1}^J$ and any pair of different points $(x, z \neq x) \in \mathcal{T}$ such that

$$\boldsymbol{Q} = \boldsymbol{U} + \boldsymbol{E}, \quad \boldsymbol{U} = \left(\begin{array}{cc|c} k(x,x) & k(x,z) & \boldsymbol{k}_J(x)^\top \\ k(z,x) & k(z,z) & \boldsymbol{k}_J(z)^\top \\ \hline \boldsymbol{k}_J(x) & \boldsymbol{k}_J(z) & \boldsymbol{K}_J \end{array}\right), \quad \boldsymbol{E} = \mathrm{diag}\big(0, 0, w, \dots, w\big).$$

If $k(\cdot, \cdot) \in \mathcal{F}_{\mathcal{M}^{-1}}$, then $\boldsymbol{U} \in \mathcal{M}^{-1}$ and $\boldsymbol{Q} = \boldsymbol{U} + \boldsymbol{E} \in \mathcal{M}^{-1}$ because of the additive diagonal closure of inverse M-matrices (Theorem 3 in [8]). Applying the second inequality in Lemma 1 to $\boldsymbol{Q}$ leads to the relation: $0 \leq k(x, z) - \boldsymbol{k}_J(x)^\top \big(w\boldsymbol{I}_J + \boldsymbol{K}_J\big)^{-1}\boldsymbol{k}_J(z) \in \mathcal{F}_{\mathcal{M}^{-1}}$ for any pair of points $(x, z)$, which completes the proof. ∎

Gaussian Cox processes (GCPs) are the gold standard for intensity estimation, and the intensity estimator (8) is the MAP solution of the permanental process [18], a variant of GCP where the square root of the intensity function is assumed to be generated from a Gaussian process. In the literature on GCPs, the advantage of permanental processes over other GCPs has been considered as the efficient estimation algorithm, and the equivalent kernels constructed by inverse M-kernels may improve the predictive performance of the fast-to-compute permanental processes by weakening the coefficient constraints.

### 3.3 Construction of Inverse M-Kernels

In the former sections, we defined a new class of kernel or inverse M-kernel, and showed some beneficial results obtained from its unique properties. Now we need to tackle a practical problem of how to construct the inverse M-kernels. Our conclusion in this paper is that for one-dimensional input space ($\mathcal{X} \subseteq \mathbb{R}$), we can find some strict inverse M-kernels, which include a well-known universal kernel called exponential/Abel/Laplace kernel; For a multi-dimensional input space, we can find some inverse M-kernels, but strict ones have yet to be discovered. In the following sections, we focus on the scenario of a one-dimensional input space, which is followed by discussions of issues with and perspectives on multi-dimensional input setting.

**Corollary 1** (Examples of strict inverse M-kernels). *Exponential kernel $k_{exp}(x, x') = e^{-|x-x'|/\tau}$ and intersection kernel $k_{int}(x, x') = \min(x, x') - \gamma$, defined on one-dimensional space $x, x' \in \mathbb{R}$, are strict inverse M-kernels. Here, $\tau$ and $\gamma$ are the hyperparameters of exponential and intersection kernels, respectively.*

*Proof.* Given a set of points $(x_1, \dots, x_N)$ sorted in ascending order $x_n < x_{n'}$ for $n < n'$, the inverse gram matrices of exponential and intersection kernels, denoted by $\boldsymbol{K}_{\exp}^{-1}$ and $\boldsymbol{K}_{\mathrm{int}}^{-1}$, respectively, are of tridiagonal form:

$$(\boldsymbol{K}_{\exp}^{-1})_{nn'} = \begin{cases} p_{n-1}^n \, p_n^{n+1}/p_{n-1}^{n+1} & : |n-n'| = 0 \\ -\sqrt{p_n^{n'}(p_n^{n'}-1)} & : |n-n'| = 1 \\ 0 & : |n-n'| > 1 \end{cases}, \quad p_n^{n'} = \begin{cases} \frac{1}{1-e^{-2|x_n-x_{n'}|/\tau}} & : n, n' \in \mathbb{N}_N \\ 1 & : \text{otherwise} \end{cases},$$

$$(\boldsymbol{K}_{\mathrm{int}}^{-1})_{nn'} = \begin{cases} q_{n-1}^n \, q_n^{n+1}/q_{n-1}^{n+1} & : |n-n'| = 0 \\ -q_n^{n+1} & : |n-n'| = 1 \\ 0 & : |n-n'| > 1 \end{cases}, \quad q_n^{n'} = \begin{cases} \frac{1}{x_{n'}-x_n} & : n, n' \in \mathbb{N}_N \\ \frac{1}{x_{n'}-\gamma} & : n = 0 \\ 1 & : n' = N+1 \end{cases},$$

Table 1: Results on KdV data across 100 trials with standard errors. $l^2$ is the integrated squared error between the approximator and the ground truth, and $cpu$ is the CPU time in second.

| | NCM | | QNM | | Our Model | |
|---|---|---|---|---|---|---|
| $\sigma$ | $l^2$ | $cpu(sec)$ | $l^2$ | $cpu(sec)$ | $l^2$ | $cpu(sec)$ |
| 0.1 | $.078 \pm .043$ | $.002 \pm .002$ | $.034 \pm .011$ | $.634 \pm .409$ | $.047 \pm .012$ | $.002 \pm .001$ |
| 0.01 | $.074 \pm .044$ | $.001 \pm .001$ | $.002 \pm .002$ | $3.49 \pm 1.04$ | $.011 \pm .001$ | $.003 \pm .001$ |

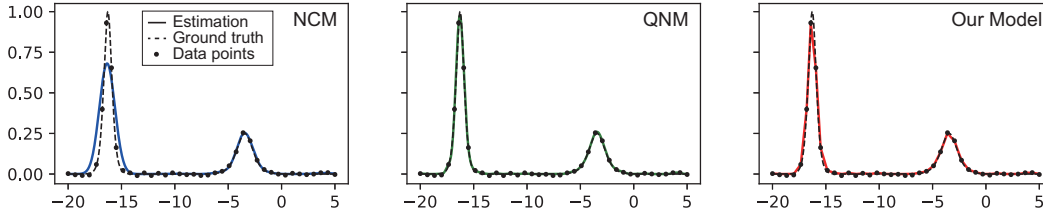

Figure 1: Estimated non-negative functions on KdV data with small noise, $\sigma = 0.01$.

where $\mathbb{N}_N = \{1, 2, \ldots, N\}$, $\tau > 0$, and $\gamma < x_1$. The results show that $\boldsymbol{K}_{\exp}^{-1}$ and $\boldsymbol{K}_{\mathrm{int}}^{-1}$ have non-positive off-diagonal entries while their inverses are entry-wise non-negative due to the corresponding kernels' properties, indicating that $\boldsymbol{K}_{\exp}, \boldsymbol{K}_{\mathrm{int}} \in \mathcal{M}^{-1}$ for any $\{x_n\}_{n=1}^N$. Therefore, $k_{\exp}, k_{\mathrm{int}} \in \mathcal{F}_{\mathcal{M}^{-1}}$. ∎

Because exponential kernel $k_{\exp}(\cdot, \cdot)$ is a universal kernel as well as a strict inverse M-kernel, Theorem 2 suggests that an inverse M-kernel model with $k_{\exp}(\cdot, \cdot)$ constitutes a linear universal approximator for one-dimensional input spaces. It should be emphasized here that more popular kernels such as Gaussian and Matérn kernels are not inverse M-kernels, and thus they cannot be used to construct linear universal approximators with non-negativity constraints, which highlights an important benefit of $k_{\exp}(\cdot, \cdot)$ that has been overlooked in the literature. However, linear models with $k_{\exp}(\cdot, \cdot)$ are generally not smooth at data points, which is a possible disadvantage given that conventional nonlinear models can employ smooth kernels. It remains to be clarified whether there exists an inverse M-kernel that is more smooth than $k_{\exp}(\cdot, \cdot)$.

It is easily verified that linear models with intersection kernels [16] constitute piece-wise linear splines. By exploiting the fact that piece-wise linear splines, whose finite set of knot values are non-negative, have non-negative values globally, Maatouk and Bay [14] proposed a Gaussian process (or equivalently a kernel method) model with non-negativity constraints, which our result re-confirms from the perspective of (strict) inverse M-kernels.

We derived the tridiagonal formulae in Corollary 1 by using some known properties of symmetric Toeplitz matrices (e.g., see [33]) as a reference. It is clear that the derived tridiagonal formula is a straightforward generalization of the inverse of Toeplitz matrices, but we cannot find any references that explicitly mention the tridiagonal formulae of one-dimensional exponential and intersection kernels.

## 4 Experiments

We examined the validity of our proposal by comparing it with conventional linear and nonlinear models on synthetic data. Here, we considered the three problems of non-negativity-constrained regression, density estimation, and intensity estimation. As benchmark models, we adopted non-negative coefficients model (NCM) in (4) and quadratic form of non-negative model (QNM) in (5) for non-negativity-constrained regression; we adopted NCM, QNM, and squared neural family (SNF) in (6) for density estimation; we adopted the intensity estimator with Gaussian kernels (IEK) [6] and the structured variational Bayesian approach with sigmoidal Gaussian Cox processes (STVB) [1] for intensity estimation. As our proposal, we adopted inverse M-kernel model (IMK) in (7) for non-negativity-constrained regression and density estimation, and intensity estimator with in-

Table 2: Results on density estimation across 100 trials with standard errors. $k_{\text{KL}}$ is the Kullback–Leibler distance between the estimation and the ground truth (the lower, the better); $cpu$ is the CPU time in seconds.

| NCM | | SNF | | QNM | | Our Model | |
|---|---|---|---|---|---|---|---|
| $d_{\text{KL}}$ | $cpu(sec)$ | $d_{\text{KL}}$ | $cpu(sec)$ | $d_{\text{KL}}$ | $cpu(sec)$ | $d_{\text{KL}}$ | $cpu(sec)$ |
| .163±.154 | .024±.007 | .452±.291 | 2.99±.970 | .163±.064 | .483±.136 | .123±.049 | .053±.021 |

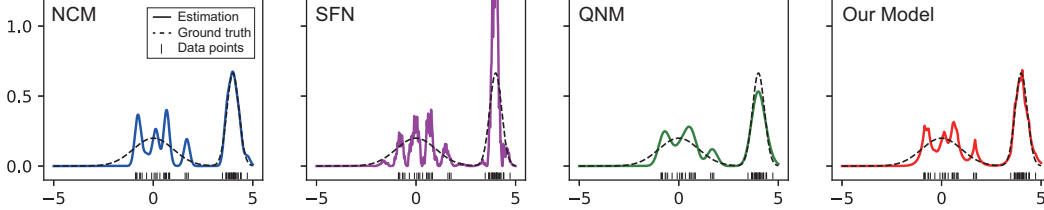

Figure 2: Estimated density functions.

verse M-kernels for intensity estimation. We employed Gaussian kernel $k(x, x') = e^{-|x-x'|^2/\tau^2}$ for NCM and QNM, and exponential kernel $k(x, x') = e^{-|x-x'|/\tau}$ for our IMK. The hyper-parameters for each model were optimized through three-fold cross validation on a grid: for NCM, QNM, and IMK, the grid is $(\tau, r) \in \mathcal{C} \otimes \mathcal{C}$ for $\mathcal{C} = \{0.1, 0.2, 0.5, 1, 2, 5, 10\}$; for SNF, the number of components for Gaussian mixture measure $d\mu(\cdot)$ was selected from $\{1, 2, 3\}$. We implemented all compared models by using Python-3.10.8 (SciPy-1.11, fnnls-1.0 (MIT License))[1]. A MacBook Pro with 12-core CPU (Apple M2 Max) was used.

## 4.1 Non-Negativity-Constrained Regression

We considered a standard regression problem with the squared loss functional, $L = \frac{1}{\sigma^2} \sum_{n=1}^{N} (y_n - f(x_n))^2$, which makes the optimization problems in (3 and A1) convex. Here $x = (x_1, \ldots, x_N)^\top$ and $y = (y_1, \ldots, y_N)^\top$ are the observed input and target values, respectively, and $\sigma^2$ is the variance of observation noise. For NCM and IMK, each of the convex problems of coefficients $\alpha$ can be recast to non-negative least squares as

$$
\begin{aligned}
\text{NCM}: \quad & \min_{\alpha \geq 0} ||C\alpha - z||^2, \quad C = [K/\sigma; \sqrt{r}U^\top], \quad z = [y/\sigma; 0_N], \\
\text{IMK}: \quad & \min_{\beta \geq 0} ||C\beta - z||^2, \quad C = [I/\sigma; \sqrt{r}U^{-1}], \quad z = [y/\sigma; 0_N], \quad \alpha = K^{-1}\beta,
\end{aligned}
\tag{10}
$$

where $U$ is the lower triangular matrix of the Cholesky decomposition of the gram matrix, $K = UU^\top$, $0_N$ is the $N$-dimensional vector with zero entries, and $[a; b]$ represents concatenation. We solved (10) with the fast nonnegative least squares [2]. For QNM, we solved the convex problem of coefficients $B$ by using the sequential least squares programming (SLSQP) [11].

In accordance with [22], we considered approximating a non-negative 2-soliton solution of the Korteweg-de Vries (KdV) equation [25], $g(x, t = 1) = \frac{12([3 + 4\cosh(2x - 8t) + \cosh(4x - 64t)])}{8[3\cosh(x - 28t) + \cosh(3x - 36t)]^2}$, where the posterior means of unconstrained Gaussian process (equivalently, kernel method regressions) tend to violate non-negativity of the function [22]. We sampled $N = 40$ data points equidistantly from the KdV solution with small noise $\sigma = 0.01$ and large noise $\sigma = 0.1$ scenarios, each of which was conducted 100 times. Then we measured the predictive performances of the models based on the integrated squared error between the result $f^*(\cdot)$ and the ground truth $g(x)$, defined as $l^2 = \int_a^b |f^*(x) - g(x)|^2 dx$ for $(a, b) = (-20, 5)$.

Table 1 displays the predictive performances achieved by the compared methods on KdV data. The comparison between the two linear models shows that our model (IMK) outperformed NCM in both noise scenarios, which is clearly illustrated by Figure 1: our model succeeded in approximating the two modes with the different scales appearing in the KdV solution, while NCM failed due to its limited representation power. See also an additional experiment in Appendix B to illustrate the difference in representation power between the two linear models. QNM achieved the best performance among the models, because the underlying function is smooth and consistent with

Table 3: Results on intensity estimation across 100 trials with standard errors. $l^2$ is the integrated squared error between the approximator and the ground truth, and $cpu$ is the CPU time in seconds.

| IEK | | STVB | | Our Model | |
|---|---|---|---|---|---|
| $l^2(\times 10^3)$ | $cpu(sec)$ | $l^2(\times 10^3)$ | $cpu(sec)$ | $l^2(\times 10^3)$ | $cpu(sec)$ |
| $7.50 \pm 3.06$ | $6.36 \pm 3.46$ | $1.74 \pm 0.504$ | $1006 \pm 54.5$ | $2.01 \pm 0.678$ | $4.53 \pm 3.15$ |

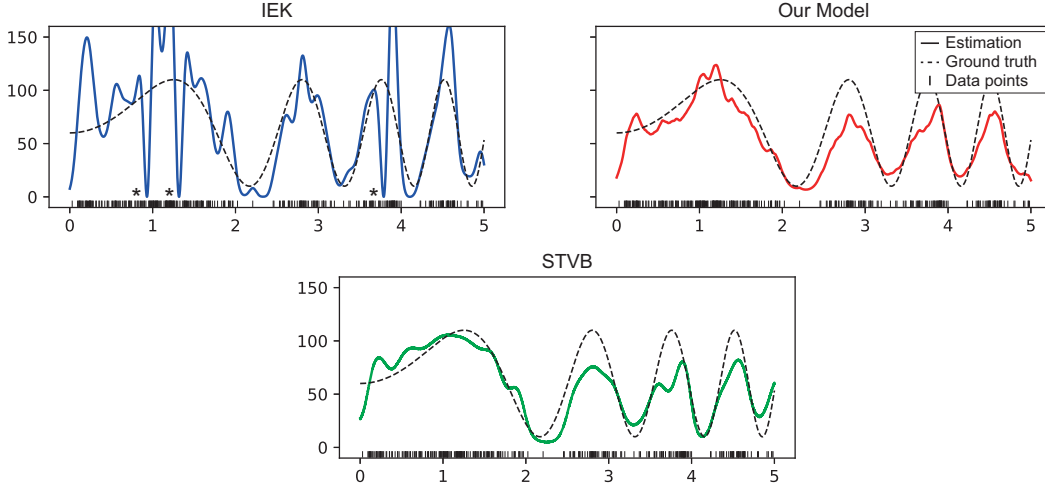

Figure 3: Estimated intensity functions. Asterisks * represent nodal points.

the Gaussian kernel that QNM adopted. The difference in performance between QNM and our model tends to shrink when observations are noisy. The advantage of our method over QNM is its computation efficiency: the linearity of the model can be fully exploited to achieve learning that is hundreds of times faster than QNM.

## 4.2 Density Estimation

We considered a density estimation problem with the loss functional, $L = -\sum_{n=1}^{N} \log f(x_n)$, where $x = (x_1, \ldots, x_N)^\top$ are the observed samples and the optimization problems in (3 and A1) are convex. Approximators of density functions are required to satisfy the normalization condition, which can be recast as a linear constraint in the linear models (NCM and IMK): $h^\top \alpha = 1$ for $(h)_n = \int_{\mathcal{X}} k(x, x_n)dx$. We trained NCM, SNF, QNM, and IMK by using SLSQP [11].

We created 100 sets of 50 samples generated from a Gaussian mixture model: $g(x) = 0.5[\mathcal{N}(x|0, 1) + \mathcal{N}(x|4, 0.3)]$, where $\mathcal{N}(\cdot|a, b)$ represents a normal distribution with mean $a$ and standard deviation $b$. The predictive performances were evaluated using the Kullback–Leibler distance (the lower, the better) between the result $f^*(x)$ and the ground truth $g(x)$, defined as $d_{\mathrm{KL}} = \int_a^b g(x) \log(g(x)/f^*(x))dx$ for $(a, b) = (-5, 5)$. Table 2 lists the results, which show that our IMK achieved better performance than NCM and comparable performance while being substantially faster than QNM. Table 2 also shows that SNF did not perform well, which might be due to overfitting to the training data, as illustrated by Figure 2. This time we assumed a small training data set ($N = 50$), where neural network-based models are likely to overfit. More careful tuning of the hyperparameters would improve SNF's performances, while the robustness against data size is generally a great advantage of kernel methods.

## 4.3 Intensity Estimation

We considered an intensity estimation problem (8), where SLSQP [11] was used to optimize IEK and our model. We implemented STVB with the TensorFlow code [1], where the number of inducing points was set as regularly aligned 100 points within the observation domain. We created 100 sets of event sequences generated from the following intensity function: $\lambda(x) = 50 \sin^2(x) + 60$

for $x \in [0, 5]$, where John and Hensman [7] reported that the nodal line problem was likely to happen. The predictive performances were evaluated using the integrated squared error between the result $\lambda^*(x)$ and the ground truth $\lambda(x)$, defined as $l^2 = \int_a^b |\lambda^*(x) - \lambda(x)|^2 dx$ for $(a, b) = (0, 5)$. Table 3 lists the results which show that our model with an equivalent inverse M-kernel achieved better performance than the naive intensity estimator with Gaussian kernel (IEK) and comparable performance to STVB while being substantially faster. Figure 3 illustrates that IEK allowed some artificial zero crossings of $\sqrt{\lambda^*(x)}$, while our model did not.

## 5  Discussions

We have proposed a novel class of kernel function, called *inverse M-kernel function*, with which we may construct flexible and linear approximators of non-negative functions. We showed that exponential kernels, which are known as universal kernels, are inverse M-kernel functions, and they can construct linear universal approximators of non-negative functions for one-dimensional input settings. We confirmed the potential benefits of our proposal experimentally on three problems: non-negative function regression, density estimation, and intensity estimation.

**Future Work and Limitations**  To the best of our knowledge, this study is the first to clarify the existence of linear universal approximators of non-negative functions, although the result is limited to one-dimensional input spaces. Constructing linear and flexible (or universal, if possible) approximators with non-negativity constraints for multi-dimensional input space is equivalent to finding inverse M-kernels for multi-dimensional input spaces, the difficulty of which can be exemplified as follows. Let $k_0 : \mathbb{R} \times \mathbb{R} \to \mathbb{R}$ be a positive semi-definite kernel, and consider the construction of a kernel for a two-dimensional input space by a popular multiplicative approach: $k(z, z') = k_0(x, x')k_0(y, y')$ for $z = (x, y)$. A gram matrix of $k(\cdot, \cdot)$, denoted by $\boldsymbol{K}$, evaluated over a Cartesian grid of input locations, $(x_1, \cdots, x_{n_x}) \otimes (y_1, \cdots, y_{n_y})$, will give rise to a matrix that can be written as the Kronecker product of two smaller gram matrices, each of which are formed by evaluating $k_0(\cdot, \cdot)$ over each input location [24]: $\boldsymbol{K} = \boldsymbol{K}_x \otimes \boldsymbol{K}_y$. If $k_0(\cdot, \cdot)$ is a strict inverse M-kernel function $k_0 \in \mathcal{F}_{\mathcal{M}^{-1}}$, then $\boldsymbol{K}_x^{-1}, \boldsymbol{K}_y^{-1} \in \mathcal{M}$, but $\boldsymbol{K}^{-1} = \boldsymbol{K}_x^{-1} \otimes \boldsymbol{K}_y^{-1} \notin \mathcal{M}$, that is, $k(z, z') \notin \mathcal{F}_{\mathcal{M}^{-1}}$: For example, some off-diagonal entries of $\boldsymbol{K}^{-1}$ are non-negative, $(\boldsymbol{K}^{-1})_{1(n_x+2)} = (\boldsymbol{K}_x^{-1})_{12}(\boldsymbol{K}_y^{-1})_{12} \geq 0$.

A possible solution to the above difficulty is to select a scalar $\eta$ large enough to satisfy the condition of inverse M-kernel (Definition 1): $\boldsymbol{K} + \eta \boldsymbol{I} \in \mathcal{M}^{-1}$. For general kernel functions, the condition is not always satisfied even under a very large $\eta$. However, if gram matrix $\boldsymbol{K}$ satisfies a specific condition called *strict path product condition* [9], then a lower bound of $\eta$ (i.e., $s(N)$ in Definition 1) that satisfies $\boldsymbol{K} + \eta \boldsymbol{I} \in \mathcal{M}^{-1}$ can be evaluated as follows.

**Theorem** (Theorem 4 in [9]). *Let $\boldsymbol{A} = (a_{ij})$ be an $n$-by-$n$ entry-wise non-negative matrix with normalized unit diagonals, $n \geq 3$. Then $\boldsymbol{A} + \eta \boldsymbol{I} \in \mathcal{M}^{-1}$ for all $\eta \geq n - 3$ if $\boldsymbol{A}$ satisfies the strict path product condition, $a_{ij}a_{jk} < a_{ik}$, for all distinct indices $i, j, k$ such that $1 \leq i, j, k \leq n$.*

Actually, it is easily verified that gram matrices of multiplicative exponential kernels, $k_{\exp}(x, s) = \prod_d e^{-|x^d - s^d|/\tau_d}$, satisfy the strict path product condition regardless of the dimensionality of the input space. Therefore, the multiplicative exponential kernels are inverse M-kernels with $s(N) = N - 3$, that is, $k_{\exp}(\cdot, \cdot) \in \mathcal{F}_{\mathcal{M}^{-1}}^{N-3}$, for multi-dimensional input spaces, which suggests that the following inverse M-kernel model (IMK) is valid:

$$f_{\text{IMK}}(x) = \sum_{n=1}^{N} \alpha_n k(x_n, x) = \boldsymbol{k}(x)^\top \alpha, \quad (\boldsymbol{K} + (N-2)\boldsymbol{I})\alpha \geqslant 0, \quad k(x, s) = \prod_{d=1}^{D} e^{-|x^d - s^d|/\tau_d}, \quad (11)$$

where $D \geq 1$ is the input dimensionality. However, as discussed in Section 3.1, the discrepancy between NCM and IMK becomes small if $s(N+1)$ for the condition $(\boldsymbol{K} + s(N+1)\boldsymbol{I})\alpha \geqslant 0$ is large, and thus $s(N+1) = N - 2$ implies that improvements of IMK (11) against NCM (4) should exist but are likely to be marginal for $N \gtrsim 10$. Because the lower bound $\eta \geq N - 3$ in the theorem above is not tight, a pressing need is to develop a method to find a smaller value of $\eta$ satisfying $(\boldsymbol{K} + \eta \boldsymbol{I}) \in \mathcal{M}^{-1}$ given kernel $k(\cdot, \cdot)$ and input points $(x_1, \ldots, x_N)$.

## Footnotes

[1]Code and data to reproduce the results are available at `https://github.com/HidKim/IM-Kernel`.

# References

[1] Virginia Aglietti, Edwin V. Bonilla, Theodoros Damoulas, and Sally Cripps. Structured variational inference in continuous Cox process models. In *Advances in Neural Information Processing Systems 32*, 2019.

[2] Rasmus Bro and Sijmen De Jong. A fast non-negativity-constrained least squares algorithm. *Journal of Chemometrics: A Journal of the Chemometrics Society*, 11(5):393–401, 1997.

[3] Jean-Paul Chiles and Pierre Delfiner. *Geostatistics: Modeling Spatial Uncertainty*, volume 713. John Wiley & Sons, 2012.

[4] David Collett. *Modelling Survival Data in Medical Research*. Chapman and Hall/CRC, 2023.

[5] Sébastien Da Veiga and Amandine Marrel. Gaussian process modeling with inequality constraints. In *Annales de la Faculté des sciences de Toulouse: Mathématiques*, volume 21, pages 529–555, 2012.

[6] Seth Flaxman, Yee Whye Teh, and Dino Sejdinovic. Poisson intensity estimation with reproducing kernels. In *Artificial Intelligence and Statistics*, pages 270–279. PMLR, 2017.

[7] S. T. John and James Hensman. Large-scale Cox process inference using variational Fourier features. In *International Conference on Machine Learning*, volume 80, pages 2362–2370. PMLR, 2018.

[8] Charles R. Johnson. Inverse M-matrices. *Linear Algebra and its Applications*, 47:195–216, 1982.

[9] Charles R. Johnson and Ronald L. Smith. Positive, path product, and inverse M-matrices. *Linear Algebra and Its Applications*, 421(2-3):328–337, 2007.

[10] Hideaki Kim, Taichi Asami, and Hiroyuki Toda. Fast Bayesian estimation of point process intensity as function of covariates. In *Advances in Neural Information Processing Systems 35*, 2022.

[11] Dieter Kraft. A software package for sequential quadratic programming. *Forschungsbericht-Deutsche Forschungs- und Versuchsanstalt fur Luft- und Raumfahrt*, 1988.

[12] Miao Liu, Girish Chowdhary, Bruno Castra Da Silva, Shih-Yuan Liu, and Jonathan P. How. Gaussian processes for learning and control: A tutorial with examples. *IEEE Control Systems Magazine*, 38(5):53–86, 2018.

[13] Weifeng Liu, Jose C. Principe, and Simon Haykin. *Kernel Adaptive Filtering: A Comprehensive Introduction*. John Wiley & Sons, 2011.

[14] Hassan Maatouk and Xavier Bay. Gaussian process emulators for computer experiments with inequality constraints. *Mathematical Geosciences*, 49:557–582, 2017.

[15] Emilio Tanowe Maddalena, Paul Scharnhorst, and Colin N. Jones. Deterministic error bounds for kernel-based learning techniques under bounded noise. *Automatica*, 134:109896, 2021.

[16] Subhransu Maji, Alexander C. Berg, and Jitendra Malik. Classification using intersection kernel support vector machines is efficient. In *2008 IEEE Conference on Computer Vision and Pattern Recognition*, pages 1–8. IEEE, 2008.

[17] Ulysse Marteau-Ferey, Francis Bach, and Alessandro Rudi. Non-parametric models for non-negative functions. In *Advances in Neural Information Processing Systems 33*, 2020.

[18] Peter McCullagh and Jesper Møller. The permanental process. *Advances in Applied Probability*, 38(4):873–888, 2006.

[19] Charles A. Micchelli, Yuesheng Xu, and Haizhang Zhang. Universal kernels. *Journal of Machine Learning Research*, 7(12), 2006.

[20] Radford M. Neal. *Bayesian Learning for Neural Networks*, volume 118. Springer Science & Business Media, 2012.

[21] John Ashworth Nelder and Robert W.M. Wedderburn. Generalized linear models. *Journal of the Royal Statistical Society Series A: Statistics in Society*, 135(3):370–384, 1972.

[22] Andrew Pensoneault, Xiu Yang, and Xueyu Zhu. Nonnegativity-enforced Gaussian process regression. *Theoretical and Applied Mechanics Letters*, 10(3):182–187, 2020.

[23] Robert J. Plemmons. M-matrix characterizations. I–nonsingular M-matrices. *Linear Algebra and its Applications*, 18(2):175–188, 1977.

[24] Yunus Saatçi. *Scalable Inference for Structured Gaussian Process Models*. PhD thesis, University of Cambridge, 2011.

[25] Nicolas Schalch. The Korteweg-de Vries Equation. *Proseminar: Algebra, Topology and Group Theory in Physics*, 2018.

[26] Bernhard Scholkopf and Alexander J. Smola. *Learning with Kernels: Support Vector Machines, Regularization, Optimization, and Beyond*. MIT press, 2018.

[27] Bernhard Schölkopf, Ralf Herbrich, and Alex J. Smola. A generalized representer theorem. In *International Conference on Computational Learning Theory*, pages 416–426. Springer, 2001.

[28] Bernhard Schölkopf, Koji Tsuda, and Jean-Philippe Vert. *Kernel Methods in Computational Biology*. MIT press, 2004.

[29] John Shawe-Taylor and Nello Cristianini. *Kernel Methods for Pattern Analysis*. Cambridge University Press, 2004.

[30] Bharath K. Sriperumbudur, Kenji Fukumizu, and Gert R.G. Lanckriet. Universality, characteristic kernels and RKHS embedding of measures. *Journal of Machine Learning Research*, 12 (7), 2011.

[31] Laura P. Swiler, Mamikon Gulian, Ari L. Frankel, Cosmin Safta, and John D. Jakeman. A survey of constrained Gaussian process regression: Approaches and implementation challenges. *Journal of Machine Learning for Modeling and Computing*, 1(2), 2020.

[32] Hiroyuki Takeda, Sina Farsiu, and Peyman Milanfar. Kernel regression for image processing and reconstruction. *IEEE Transactions on Image Processing*, 16(2):349–366, 2007.

[33] William F. Trench. Properties of some generalizations of Kac-Murdock-Szegö Matrices. *Contemporary Mathematics*, 281:233–245, 2001.

[34] Russell Tsuchida, Cheng Soon Ong, and Dino Sejdinovic. Squared neural families: A new class of tractable density models. In *Advances in Neural Information Processing Systems 36*, 2023.

[35] Grace Wahba. *Spline Models for Observational Data*, volume 59. SIAM, 1990.

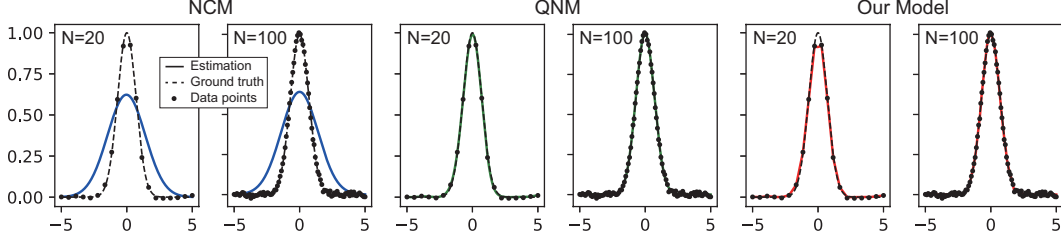

Figure B1: Estimation results of regression problem on ground truth $g(x) = e^{-|x|^2}$ by using reference models (NCM and QNM) with Gaussian kernel $e^{-|x-x'|^2/4}$ and our model with inverse M-kernel $e^{-|x-x'|/2}$.

# A   Details of QNM

Marteau-Ferey et al. [17] considered the following problem of positive semi-definite operator $A$,

$$\inf_A L(f_A(x_1), \ldots, f_A(x_N)) + r||A||_* + r_2||A||_F^2, \tag{A1}$$

where $||\cdot||_*$ and $||\cdot||_F^2$ are the nuclear norm and the squared Frobenius norm, respectively, and $f_A(\cdot)$ is defined as $f_A(\cdot) = \phi(\cdot)^\top A \phi(\cdot)$ for $\phi(\cdot)$ the feature map of $A$. Then they showed that the solution of (A1) holds for a representer theorem, which leads to the quadratic form of approximator (5). The coefficient matrix $B$ in (5) is obtained efficiently by solving the $N$-dimensional dual problem,

$$\alpha^* = \arg \sup_{\alpha \in \mathbb{R}^N} -L^*(\alpha) - \frac{1}{2r_2}||\big[V \mathrm{diag}(\alpha)V^\top + rI\big]_-||_F^2, \tag{A2}$$

$$B = r_2^{-1}V^{-1}\big[V \mathrm{diag}(\alpha^*)V^\top + rI\big]_- V^{-\top}, \tag{A3}$$

where $[A]_-$ represents the negative part of $A$ (for details, see [17]), $V$ is the Cholesky decomposition of $K := [k(x_n, x_{n'})]_{nn'}$, i.e., $K = V^\top V$, $L^*(\alpha) = \sum_{n=1}^N l^*(\alpha_n)$ represents the Fenchel conjugate of the loss functional $L(z_1, \ldots, z_N) = \sum_{n=1}^N l(z_n)$. More concretely, $l^*(\alpha_n) = y_n \alpha_n + \frac{1}{2}\sigma^2 \alpha_n^2$ for $l(z_n) = \frac{1}{2\sigma^2}(z_n - y_n)^2$, and $l^*(\alpha_n) = -(1 + \log(-\alpha_n))$ for $l(z_n) = -\log(z_n)$.

# B   Additional Results

To clearly discern the difference in representation power between the two linear models, NCM and IMK, we conducted experiments on $N = 20$ and $N = 100$ data points sampled equidistantly from $g(x) = e^{-|x|^2}$ with noise $\sigma = 0.01$, where we set the scale parameter $\tau$ of kernel function to be twice as large as the ground truth, $\tau = 2$, for all models. Figure B1 displays the results: NCM failed to recover the functional form of the ground truth even with a large number of training points, while our IMK, which invokes the universal approximation under a strict inverse M-kernel function, achieved good estimation results. As emphasized by Marteau-Ferey et al. [17], NCM cannot approximate a function with a scale strictly smaller than the used kernel's scale well, which raises a problematic trade-off when the underlying function has different scales of components: adjusting $\tau$ to the smaller scale components causes overfitting in the larger scale ones, while adjusting $\tau$ to the larger scale ones fails to recover the smaller scale ones. The experiments in Section 4.1 replicated this problematic situation.

